# Sufficient Conditions for Agnostic Active Learnable

**Liwei Wang**
Key Laboratory of Machine Perception, MOE,
School of Electronics Engineering and Computer Science,
Peking University,
wanglw@cis.pku.edu.cn

## Abstract

We study pool-based active learning in the presence of noise, i.e. the agnostic setting. Previous works have shown that the effectiveness of agnostic active learning depends on the learning problem and the hypothesis space. Although there are many cases on which active learning is very useful, it is also easy to construct examples that no active learning algorithm can have advantage. In this paper, we propose intuitively reasonable sufficient conditions under which agnostic active learning algorithm is strictly superior to passive supervised learning. We show that under some noise condition, if the Bayesian classification boundary and the underlying distribution are smooth to a finite order, active learning achieves polynomial improvement in the label complexity; if the boundary and the distribution are infinitely smooth, the improvement is exponential.

## 1 Introduction

Active learning addresses the problem that the algorithm is given a pool of unlabeled data drawn i.i.d. from some underlying distribution. The algorithm can then pay for the label of any example in the pool. The goal is to learn an accurate classifier by requesting as few labels as possible. This is in contrast with the standard passive supervised learning, where the labeled examples are chosen randomly.

The simplest example that demonstrates the potential of active learning is to learn the optimal threshold on an interval. If there exists a perfect threshold separating the two classes (i.e. there is no noise), then binary search only needs $O(\ln \frac{1}{\epsilon})$ labels to learn an $\epsilon$-accurate classifier, while passive learning requires $O(\frac{1}{\epsilon})$ labels. Another encouraging example is to learn a homogeneous linear separator for data uniformly distributed on the unit sphere of $\mathbb{R}^d$. In this case active learning can still give exponential savings in the label complexity [Das05].

However, there are also very simple problems that active learning does not help at all. Suppose the instances are uniformly distributed on $[0, 1]$, and the positive class could be any interval on $[0, 1]$. Any active learning algorithms needs $O(\frac{1}{\epsilon})$ label requests to learn an $\epsilon$-accurate classifier [Han07]. There is no improvement over passive learning. All above are noise-free (realizable) problems. Of more interest and more realistic is the agnostic setting, where the class labels can be noisy so that the best classifier in the hypothesis space has a non-zero error $\nu$. For agnostic active learning, there is no active learning algorithm that can always reduce label requests due to a lower bound $\Omega(\frac{\nu^2}{\epsilon^2})$ for the label complexity [Kaa06].

It is known that whether active learning helps or not depends on the distribution of the instance-label pairs and the hypothesis space. Thus a natural question would be that under what conditions is active learning guaranteed to require fewer labels than passive learning.

In this paper we propose intuitively reasonable sufficient conditions under which active learning achieves lower label complexity than that of passive learning. Specifically, we focus on the $A^2$ algorithm [BAL06] which works in the agnostic setting. Earlier work has discovered that the label complexity of $A^2$ can be upper bounded by a parameter of the hypothesis space and the data distribution called *disagreement coefficient* [Han07]. This parameter often characterizes the intrinsic difficulty of the learning problem. By an analysis of the disagreement coefficient we show that, under some noise condition, if the Bayesian classification boundary and the underlying distribution are smooth to a finite order, then $A^2$ gives polynomial savings in the label complexity; if the boundary and the distribution are infinitely smooth, $A^2$ gives exponential savings.

## 1.1 Related Works

Our work is closely related to [CN07], in which the authors proved sample complexity bounds for problems with smooth classification boundary under Tsybakov's noise condition [Tsy04]. They also assumed that the distribution of the instances is bounded from above and below. The main difference to our work is that their analysis is for the membership-query setting [Ang88], in which the learning algorithm can choose any point in the instance space and ask for its label; while the pool-based model analyzed here assumes the algorithm can only request labels of the instances it observes.

Another related work is due to Friedman [Fri09]. He introduced a different notion of smoothness and showed that this guarantees exponential improvement for active learning. But his work focused on the realizable case and does not apply to the agnostic setting studied here.

Soon after $A^2$, Dasgupta, Hsu and Monteleoni [DHM07] proposed an elegant agnostic active learning algorithm. It reduces active learning to a series of supervised learning problems. If the hypothesis space has a finite VC dimension, it has a better label complexity than $A^2$. However, this algorithm relies on the *normalized* uniform convergence bound for the VC class. It is not known whether it holds for more general hypothesis space such as the smooth boundary class analyzed in this paper. (For recent advances on this topic, see [GKW03].) It is left as an open problem whether our results apply to this algorithm by refined analysis of the normalized bounds.

## 2 Preliminaries

Let $\mathcal{X}$ be an instance space, $\mathcal{D}$ a distribution over $\mathcal{X} \times \{-1, 1\}$. Let $\mathcal{H}$ be the hypothesis space, a set of classifiers from $\mathcal{X}$ to $\{\pm 1\}$. Denote $\mathcal{D}_{\mathcal{X}}$ the marginal of $\mathcal{D}$ over $\mathcal{X}$. In our active learning model, the algorithm has access to a pool of unlabeled examples from $\mathcal{D}_{\mathcal{X}}$. For any unlabeled point $x$, the algorithm can ask for its label $y$, which is generated from the conditional distribution at $x$. The error of a hypothesis $h$ according to $\mathcal{D}$ is $er_{\mathcal{D}}(h) = \Pr_{(x,y)\sim\mathcal{D}}(h(x) \neq y)$. The empirical error on a finite sample $\mathcal{S}$ is $er_{\mathcal{S}}(h) = \frac{1}{|\mathcal{S}|}\sum_{(x,y)\in\mathcal{S}} \mathbb{I}[h(x) \neq y]$, where $\mathbb{I}$ is the indicator function. We use $h^*$ denote the best classifier in $\mathcal{H}$. That is, $h^* = \arg\min_{h\in\mathcal{H}} er_{\mathcal{D}}(h)$. Let $\nu = er_{\mathcal{D}}(h^*)$. Our goal is to learn a $\hat{h} \in \mathcal{H}$ with error rate at most $\nu + \epsilon$, where $\epsilon$ is a predefined parameter.

$A^2$ is the first rigorous agnostic active learning algorithm. A description of the algorithm is given in Fig.1. It was shown that $A^2$ is never much worse than passive learning in terms of the label complexity. The key observation that $A^2$ can be superior to passive learning is that, since our goal is to choose an $\hat{h}$ such that $er_{\mathcal{D}}(\hat{h}) \leq er_{\mathcal{D}}(h^*) + \epsilon$, we only need to *compare* the errors of hypotheses. Therefore we can just request labels of those $x$ on which the hypotheses under consideration have disagreement.

To do this, the algorithm keeps track of two spaces. One is the current version space $V_i$, consisting of hypotheses that with statistical confidence are not too bad compared to $h^*$. To achieve such a statistical guarantee, the algorithm must be provided with a uniform convergence bound over the hypothesis space. That is, with probability at least $1 - \delta$ over the draw of sample $\mathcal{S}$ according to $\mathcal{D}$,
$$LB(\mathcal{S}, h, \delta) \leq er_{\mathcal{D}}(h) \leq UB(\mathcal{S}, h, \delta),$$
hold simultaneously for all $h \in \mathcal{H}$, where the lower bound $LB(\mathcal{S}, h, \delta)$ and upper bound $UB(\mathcal{S}, h, \delta)$ can be computed from the empirical error $er_{\mathcal{S}}(h)$. The other space is the region of disagreement $DIS(V_i)$, which is the set of all $x \in \mathcal{X}$ for which there are hypotheses in $V_i$ that disagree on $x$. Formally, for any $V \subset \mathcal{H}$,
$$DIS(V) = \{x \in \mathcal{X} : \exists h, h' \in V, \ h(x) \neq h'(x)\}.$$

```
Input: concept space $\mathcal{H}$, accuracy parameter $\epsilon \in (0,1)$, confidence parameter $\delta \in (0,1)$;
Output: classifier $\hat{h} \in \mathcal{H}$;
Let $\hat{n} = 2(2\log_2 \frac{\lambda}{\epsilon} + \ln \frac{1}{\delta})\log_2 \frac{2}{\epsilon}$   ($\lambda$ depends on $\mathcal{H}$ and the problem, see Theorem 5) ;
Let $\delta' = \delta/\hat{n}$ ;
$V_0 \leftarrow \mathcal{H}$, $S_0 \leftarrow \emptyset$, $i \leftarrow 0$, $j_1 \leftarrow 0$, $k \leftarrow 1$ ;
while $\Delta(V_i)(\min_{h\in V_i} UB(S_i,h,\delta') - \min_{h\in V_i} LB(S_i,h,\delta')) > \epsilon$ do
    $V_{i+1} \leftarrow \{h \in V_i : LB(S_i,h,\delta') \le \min_{h'\in V_i} UB(S_i,h',\delta')\}$;
    $i \leftarrow i+1$;
    if $\Delta(V_i) < \frac{1}{2}\Delta(V_{j_k})$ then
        $k \leftarrow k+1$; $j_k \leftarrow i$;
    end
    $S_i' \leftarrow$ Rejection sample $2^{i-j_k}$ samples $x$ from $D$ satisfying $x \in DIS(V_i)$;
    $S_i \leftarrow \{(x,y = label(x)) : x \in S_i'\}$;
end
Return $\hat{h} = \operatorname{argmin}_{h\in V_i} UB(S_i,h,\delta')$.
```

**Algorithm 1**: The $A^2$ algorithm (this is the version in [Han07])

The volume of $DIS(V)$ is denoted by $\Delta(V) = \Pr_{X\sim\mathcal{D}_\mathcal{X}}(X \in DIS(V))$. Requesting labels of the instances from $DIS(V_i)$ allows $A^2$ require fewer labels than passive learning. Hence the key issue is how fast $\Delta(V_i)$ reduces. This process, and in turn the label complexity of $A^2$, are nicely characterized by the disagreement coefficient $\theta$ introduced in [Han07].

**Definition 1** *Let $\rho(\cdot,\cdot)$ be the pseudo-metric on a hypothesis space $\mathcal{H}$ induced by $\mathcal{D}_\mathcal{X}$. That is, for $h, h' \in \mathcal{H}$, $\rho(h,h') = \Pr_{X\sim\mathcal{D}_\mathcal{X}}(h(X) \ne h'(X))$. Let $B(h,r) = \{h' \in \mathcal{H}: \rho(h,h') \le r\}$. The disagreement coefficient $\theta(\epsilon)$ is*

$$\theta(\epsilon) = \sup_{r\ge\epsilon} \frac{\Pr_{X\sim\mathcal{D}_\mathcal{X}}(X \in DIS(B(h^*,r)))}{r}, \tag{1}$$

*where $h^* = \arg\min_{h\in\mathcal{H}} er_\mathcal{D}(h)$.*

Note that $\theta$ depends on $\mathcal{H}$ and $\mathcal{D}$, and $1 \le \theta(\epsilon) \le \frac{1}{\epsilon}$.

## 3 Main Results

As mentioned earlier, whether active learning helps or not depends on the distribution and the hypothesis space. There are simple examples such as learning intervals for which active learning has no advantage. However, these negative examples are more or less "artificial". It is important to understand whether problems with practical interest are actively learnable or not. In this section we provide intuitively reasonable conditions under which the $A^2$ algorithm is strictly superior to passive learning. Our main results (Theorem 11 and Theorem 12) show that if the learning problem has a smooth Bayes classification boundary, and the distribution $\mathcal{D}_\mathcal{X}$ has a density bounded by a smooth function, then under some noise condition $A^2$ saves label requests. It is a polynomial improvement for finite smoothness, and exponential for infinite smoothness.

In Section 3.1 we formally define the smoothness and introduce the hypothesis space, which contains smooth classifiers. We show a uniform convergence bound of order $O(n^{-1/2})$ for this hypothesis space. This bound determines $UB(\mathcal{S},h,\delta)$ and $LB(\mathcal{S},h,\delta)$ in $A^2$. Section 3.2 is the main technical part, where we give upper bounds for the disagreement coefficient of smooth problems. In Section 3.3 we show that under some noise condition, there is a sharper bound for the label complexity in terms of the disagreement coefficient. These lead to our main results.

### 3.1 Smoothness

Let $f$ be a function defined on $\Omega \subset \mathbb{R}^d$. For any vector $\mathbf{k} = (k_1, \cdots, k_d)$ of $d$ nonnegative integers, let $|\mathbf{k}| = \sum_{i=1}^{d} k_i$. Define the $K$-norm as

$$\|f\|_K := \max_{|\mathbf{k}| \le K-1} \sup_{x \in \Omega} |D^{\mathbf{k}} f(x)| + \max_{|\mathbf{k}| = K-1} \sup_{x, x' \in \Omega} \frac{D^{\mathbf{k}} f(x) - D^{\mathbf{k}} f(x')}{\|x - x'\|}, \tag{2}$$

where

$$D^{\mathbf{k}} = \frac{\partial^{|\mathbf{k}|}}{\partial^{k_1} x_1 \cdots \partial^{k_d} x_d},$$

is the differential operator.

**Definition 2** *(Finite Smooth Functions) A function $f$ is said to be $K$th order smooth with respect to a constant $C$, if $\|f\|_K \le C$. The set of $K$th order smooth functions is defined as*

$$F_C^K := \{ f : \|f\|_K \le C \}. \tag{3}$$

Thus $K$th order smooth functions have uniformly bounded partial derivatives up to order $K-1$, and the $K-1$th order partial derivatives are Lipschitz.

**Definition 3** *(Infinitely Smooth Functions) A function $f$ is said to be infinitely smooth with respect to a constant $C$, if $\|f\|_K \le C$ for all nonnegative integers $K$. The set of infinitely smooth functions is denoted by $F_C^\infty$.*

With the definitions of smoothness, we introduce the hypothesis space we use in the $A^2$ algorithm.

**Definition 4** *(Hypotheses with Smooth Boundaries) A set of hypotheses $\mathcal{H}_C^K$ defined on $[0,1]^{d+1}$ is said to have $K$th order smooth boundaries, if for every $h \in \mathcal{H}_C^K$, the classification boundary is a $K$th order smooth function on $[0,1]^d$. To be precise, let $\mathbf{x} = (x^1, x^2, \ldots, x^{d+1}) \in [0,1]^{d+1}$. The classification boundary is the graph of function $x^{d+1} = f(x^1, \ldots, x^d)$, where $f \in F_C^K$. Similarly, a hypothesis space $\mathcal{H}_C^\infty$ is said to have infinitely smooth boundaries, if for every $h \in \mathcal{H}_C^\infty$ the classification boundary is the graph an infinitely smooth function on $[0,1]^d$.*

Previous results on the label complexity of $A^2$ assumes the hypothesis space has finite VC dimension. The goal is to ensure a $O(n^{-1/2})$ uniform convergence bound so that $UB(\mathcal{S}, h, \delta) - LB(\mathcal{S}, h, \delta) = O(n^{-1/2})$. The hypothesis space $\mathcal{H}_C^K$ and $\mathcal{H}_C^\infty$ do not have finite VC dimensions. Compared with the VC class, $\mathcal{H}_C^K$ and $\mathcal{H}_C^\infty$ are exponentially larger in terms of the covering numbers [vdVW96]. But uniform convergence bound still holds for $\mathcal{H}_C^K$ and $\mathcal{H}_C^\infty$ under a broad class of distributions. The following theorem is a consequence of some known results in empirical processes.

**Theorem 5** *For any distribution $\mathcal{D}$ over $[0,1]^{d+1} \times \{-1, 1\}$, whose marginal distribution $\mathcal{D}_\mathcal{X}$ on $[0,1]^{d+1}$ has a density upper bounded by a constant $M$, and any $0 < \delta \le \delta_0$ ($\delta_0$ is a constant), with probability at least $1 - \delta$ over the draw of the training set $\mathcal{S}$ of $n$ examples,*

$$|er_{\mathcal{D}}(h) - er_{\mathcal{S}}(h)| \le \lambda \sqrt{\frac{\log \frac{1}{\delta}}{n}}, \tag{4}$$

*holds simultaneously for all $h \in \mathcal{H}_C^K$ provided $K > d$ (or $K = \infty$). Here $\lambda$ is a constant depending only on $d$, $K$, $C$ and $M$.*

**Proof** It can be seen, from Corollary 2.7.3 in [vdVW96] that the *bracketing numbers* $N_{[\,]}$ of $\mathcal{H}_C^K$ satisfies $\log N_{[\,]}(\epsilon, \mathcal{H}_C^K, L_2(\mathcal{D}_\mathcal{X})) = O((\frac{1}{\epsilon})^{\frac{2d}{K}})$. Since $K > d$, then there exist constants $c_1, c_2$ such that

$$P_{\mathcal{D}} \left( \sup_{h \in \mathcal{H}_C^K} |er(h) - er_{\mathcal{S}}(h)| \ge t \right) \le c_1 \exp \left( -\frac{nt^2}{c_2} \right)$$

for all $nt^2 \ge t_0$, where $t_0$ is some constant (see Theorem 5.11 and Lemma 5.10 of [vdG00]). Let $\delta = c_1 \exp \left( -\frac{nt^2}{c_2} \right)$, the theorem follows. ∎

Now we can determine $UB(\mathcal{S}, h, \delta)$ and $LB(\mathcal{S}, h, \delta)$ for $A^2$ by simply letting $UB(\mathcal{S}, h, \delta) = er_{\mathcal{S}}(h) + \lambda \sqrt{\frac{\ln \frac{1}{\delta}}{n}}$ and $LB(\mathcal{S}, h, \delta) = er_{\mathcal{S}}(h) - \lambda \sqrt{\frac{\ln \frac{1}{\delta}}{n}}$, where $\mathcal{S}$ is of size $n$.

## 3.2 Disagreement Coefficient

The disagreement coefficient $\theta$ plays an important role for the label complexity of active learning algorithms. In fact previous negative examples for which active learning does not work are all the results of large $\theta$. For instance the interval learning problem, $\theta(\epsilon) = \frac{1}{\epsilon}$, which leads to the same label complexity as passive learning. In the following two theorems we show that the disagreement coefficient $\theta(\epsilon)$ for smooth problems is small.

**Theorem 6** *Let the hypothesis space be $\mathcal{H}_C^K$. If the distribution $\mathcal{D}_{\mathcal{X}}$ has a density $p(x^1, \ldots, x^{d+1})$ such that there exists a $K$th order smooth function $g(x^1, \ldots, x^{d+1})$ and two constants $0 < \alpha \leq \beta$ such that $\alpha g(x^1, \ldots, x^{d+1}) \leq p(x^1, \ldots, x^{d+1}) \leq \beta g(x^1, \ldots, x^{d+1})$ for all $(x^1, \ldots, x^{d+1}) \in [0,1]^{d+1}$, then $\theta(\epsilon) = O\left(\left(\frac{1}{\epsilon}\right)^{\frac{d}{K+d}}\right)$.*

**Theorem 7** *Let the hypothesis space be $\mathcal{H}_C^{\infty}$. If the distribution $\mathcal{D}_{\mathcal{X}}$ has a density $p(x^1, \ldots, x^{d+1})$ such that there exist an infinitely smooth function $g(x^1, \ldots, x^d)$ and two constants $0 < \alpha \leq \beta$ such that $\alpha g(x^1, \ldots, x^d) \leq p(x^1, \ldots, x^{d+1}) \leq \beta g(x^1, \ldots, x^d)$ for all $(x^1, \ldots, x^{d+1}) \in [0,1]^{d+1}$, then $\theta(\epsilon) = O(\log^d(\frac{1}{\epsilon}))$.*

The key points in the theorems are: the classification boundaries are smooth; and the density is bounded from above and below by constants times a smooth function. These two conditions include a large class of learning problems. Note that the density itself is not necessarily smooth. We just require the density does not change too rapidly.

The intuition behind the two theorems above is as follows. Let $f_{h^*}(x)$ and $f_h(x)$ be the classification boundaries of $h^*$ and $h$, and suppose $\rho(h, h^*)$ is small, where $\rho(h, h^*) = \Pr_{x \sim \mathcal{D}_{\mathcal{X}}}(h(x) \neq h^*(x))$ is the pseudo metric. If the classification boundaries and the density are all smooth, then the two boundaries have to be close to each other everywhere. That is, $|f_h(x) - f_{f^*}(x)|$ is small uniformly for all $x$. Hence only the points close to the classification boundary of $h^*$ can be in $DIS(B(h^*, \epsilon))$, which leads to a small disagreement coefficient.

The proofs of Theorem 6 and Theorem 7 rely on the following two lemmas.

**Lemma 8** *Let $\Phi$ be a function defined on $[0,1]^d$ and $\int_{[0,1]^d} |\Phi(x)| dx \leq r$. If there exists a $K$th order smooth function $\tilde{\Phi}$ and $0 < \alpha \leq \beta$ such that $\alpha|\tilde{\Phi}(x)| \leq |\Phi(x)| \leq \beta|\tilde{\Phi}(x)|$ for all $x \in [0,1]^d$, then $\|\Phi\|_{\infty} = O(r^{\frac{K}{K+d}}) = O(r \cdot (\frac{1}{r})^{\frac{d}{K+d}})$, where $\|\Phi\|_{\infty} = \sup_{x \in [0,1]^d} |\Phi(x)|$.*

**Lemma 9** *Let $\Phi$ be a function defined on $[0,1]^d$ and $\int_{[0,1]^d} |\Phi(x)| dx \leq r$. If there exists an infinitely smooth function $\tilde{\Phi}$ and $0 < \alpha \leq \beta$ such that $\alpha|\tilde{\Phi}(x)| \leq |\Phi(x)| \leq \beta|\tilde{\Phi}(x)|$ for all $x \in [0,1]^d$, then $\|\Phi\|_{\infty} = O(r \cdot \log^d(\frac{1}{r}))$*

We will briefly describe the ideas of the proofs of these two lemmas in the Appendix. The formal proofs are given in the supplementary file.

**Proof of Theorem 6** First of all, since we focus on binary classification, $DIS(B(h^*, r))$ can be written equivalently as

$$DIS(B(h^*, r)) = \{x \in \mathcal{X}, \exists h \in B(h^*, r), \; s.t. \; h(x) \neq h^*(x)\}.$$

Consider any $h \in B(h^*, r)$. Let $f_h, f_{h^*} \in F_C^K$ be the corresponding classification boundaries of $h$ and $h^*$ respectively. If $r$ is sufficiently small, we must have

$$\rho(h, h^*) = \Pr_{X \sim \mathcal{D}_{\mathcal{X}}}(h(X) \neq h^*(X)) = \int_{[0,1]^d} dx^1 \ldots dx^d \left| \int_{f_{h^*}(x^1, \ldots, x^d)}^{f_h(x^1, \ldots, x^d)} p(x^1, \ldots, x^{d+1}) dx^{d+1} \right|.$$

Denote

$$\Phi_h(x^1, \ldots, x^d) = \int_{f_{h^*}(x^1, \ldots, x^d)}^{f_h(x^1, \ldots, x^d)} p(x^1, \ldots, x^{d+1}) dx^{d+1}.$$

We assert that there is a $K$th order smooth function $\tilde{\Phi}_h(x^1, \ldots, x^d)$ and two constants $0 < u \leq v$ such that $u|\tilde{\Phi}_h| \leq |\Phi_h| \leq v|\tilde{\Phi}_h|$. To see this, remember that $f_h$ and $f_{h^*}$ are $K$th order smooth functions; and the density $p$ is upper and lower bounded by constants times a $K$th order smooth function $g(x^1, \ldots, x^{d+1})$; and note that $\tilde{\Phi}_h(x^1, \ldots, x^d) = \int_{f_{h^*}(x^1, \ldots, x^d)}^{f_h(x^1, \ldots, x^d)} g(x^1, \ldots, x^{d+1}) dx^{d+1}$ is a $K$th order smooth function. The latter is easy to check by taking derivatives. By Lemma 8, we have $\|\Phi_h\|_\infty = O(r \cdot (\frac{1}{r})^{\frac{d}{K+d}})$, since $\int |\Phi_h| = \rho(h, h^*) \leq r$. Because this holds for all $h \in B(h^*, r)$, we have $\sup_{h \in B(h^*, r)} \|\Phi_h\|_\infty = O(r \cdot (\frac{1}{r})^{\frac{d}{K+d}})$.

Now consider the region of disagreement of $B(h^*, r)$. Clearly $DIS(B(h^*, r)) = \cup_{h \in B(h^*, r)} \{x : h(x) \neq h^*(x)\}$. Hence

$$\Pr_{X \sim \mathcal{D}_\mathcal{X}} (x \in DIS(B(h^*, r))) = \Pr_{X \sim \mathcal{D}_\mathcal{X}} \left(x \in \cup_{h \in B(h^*, r)} \{x : h(x) \neq h^*(x)\}\right)$$

$$\leq 2 \sup_{h \in B(h^*, r)} \int_{[0,1]^d} \|\Phi_h\|_\infty dx^1 \ldots dx^d = O\left(r \cdot \left(\frac{1}{r}\right)^{\frac{d}{K+d}}\right).$$

The theorem follows by the definition of $\theta(\epsilon)$. ∎

Theorem 7 can be proved similarly by using Lemma 9.

### 3.3 Label Complexity

It was shown in [Han07] that the label complexity of $A^2$ is

$$O\left(\theta^2 \left(\frac{\nu^2}{\epsilon^2} + 1\right) polylog\left(\frac{1}{\epsilon}\right) \ln \frac{1}{\delta}\right), \tag{5}$$

where $\nu = \min_{h \in \mathcal{H}} er_\mathcal{D}(h)$. When $\epsilon \geq \nu$, our previous results on the disagreement coefficient already imply polynomial or exponential improvements for $A^2$. However, when $\epsilon < \nu$, the label complexity becomes $O(\frac{1}{\epsilon^2})$, the same as passive learning whatever $\theta$ is. In fact, without any assumption on the noise, the $O(\frac{1}{\epsilon^2})$ result is inevitable due to the $\Omega(\frac{\nu^2}{\epsilon^2})$ lower bound of agnostic active learning [Kaa06].

Recently, there has been considerable interest in how noise affects the learning rate. A remarkable notion is due to Tsybakov [Tsy04], which was first introduced for passive learning. Let $\eta(x) = P(Y = 1|X = x)$. Tsybakov's noise condition assumes that for some $c > 0$, $0 < \alpha \leq \infty$

$$\Pr_{X \sim \mathcal{D}_\mathcal{X}} (|\eta(X) - 1/2| \leq t) \leq ct^{-\alpha}, \tag{6}$$

for all $0 < t \leq t_0$, where $t_0$ is some constant. (6) implies a connection between the pseudo distance $\rho(h, h^*)$ and the excess risk $er_\mathcal{D}(h) - er_\mathcal{D}(h^*)$:

$$\rho(h, h^*) \leq c' \left(er_\mathcal{D}(h) - er_\mathcal{D}(h^*)\right)^{1/\kappa}, \tag{7}$$

where $h^*$ is the Bayes classifier, $c'$ is some finite constant. Here $\kappa = \frac{1+\alpha}{\alpha} \geq 1$ is called the noise exponent. $\kappa = 1$ is the optimal case, where the problem has bounded noise; $\kappa > 1$ correspond to unbounded noise.

Castro and Nowak [CN07] noticed that Tsybakov's noise condition is also important in active learning. They proved label complexity bounds in terms of $\kappa$ for the membership-query setting. A notable fact is that $\tilde{O}((\frac{1}{\epsilon})^{\frac{2\kappa-2}{\kappa}})$ $(\kappa > 1)$ is both an upper and a lower bound for membership-query in the minimax sense. It is important to point out that the lower bound automatically applies to pool-based model, since pool makes weaker assumptions than membership-query. Hence for large $\kappa$, active learning has very limited improvement over passive learning whatever other factors are.

Recently, Hanneke [Han09] obtained similar label complexity for pool-based model. He showed the labels requested by $A^2$ is $O(\theta^2 \ln \frac{1}{\epsilon} \ln \frac{1}{\delta})$ for the bounded noise case, i.e. $\kappa = 1$. Here we slightly

generalize Hanneke's result to unbounded noise by introducing the following noise condition. We assume there exist $c_1, c_2 > 0$ and $T_0 > 0$ such that

$$\Pr_{X \sim \mathcal{D}_{\mathcal{X}}}(|\eta(X) - 1/2| \leq \frac{1}{T}) \leq c_1 e^{-c_2 T}, \tag{8}$$

for all $T \geq T_0$. It is not difficult to show that (8) implies

$$\rho(h, h^*) = O\left((er(h) - er(h^*)) \ln \frac{1}{(er(h) - er(h^*))}\right). \tag{9}$$

This condition assumes unbounded noise. Under this noise condition, $A^2$ has a better label complexity.

**Theorem 10** *Assume that the learning problem satisfies the noise condition (8) and $\mathcal{D}_{\mathcal{X}}$ has a density upper bounded by a constant $M$. For any hypothesis space $\mathcal{H}$ that has a $O(n^{-1/2})$ uniform convergence bound, if the Bayes classifier $h^*$ is in $\mathcal{H}$, then with probability at least $1 - \delta$, $A^2$ outputs $\hat{h} \in \mathcal{H}$ with $er_{\mathcal{D}}(\hat{h}) \leq er_{\mathcal{D}}(h^*) + \epsilon$, and the number of labels requested by the algorithm is at most $O(\theta^2(\epsilon) \cdot \ln \frac{1}{\delta} \cdot polylog(\frac{1}{\epsilon}))$.*

**Proof** As the proof of [Han07], one can show that with probability $1 - \delta$ we never remove $h^*$ from $V_i$, and for any $h, h' \in V_i$ we must have $\Delta(V_i)(er_i(h) - er_i(h')) = er_{\mathcal{D}}(h) - er_{\mathcal{D}}(h')$, where $er_i(h)$ is the error rate of $h$ conditioned on $DIS(V_i)$. These guarantees $er_{\mathcal{D}}(\hat{h}) \leq er_{\mathcal{D}}(h^*) + \epsilon$.

If $\Delta(V_i) \leq 2\epsilon\theta(\epsilon)$, due to the $O(n^{-1/2})$ uniform convergence bound, $O(\theta^2(\epsilon) \ln \frac{1}{\delta})$ labels suffices to make $\Delta(V_i)(UB(\mathcal{S}_i, h, \delta') - LB(\mathcal{S}_i, h, \delta')) \leq \epsilon$ for all $h \in DIS(V_i)$ and the algorithm stops. Hence we next consider $\Delta(V_i) > 2\epsilon\theta(\epsilon)$. Note that there are at most $O(\ln \frac{1}{\epsilon})$ times $\Delta(V_i) < \frac{1}{2}\Delta(V_{j_k})$ occurs. So below we bound the number of labels needed to make $\Delta(V_i) < \frac{1}{2}\Delta(V_{j_k})$ occurs. By the definition of $\theta(\epsilon)$, if $\rho(h, h^*) \leq \frac{\Delta(V_{j_k})}{2\theta(\epsilon)}$ for all $h \in V_i$, then $\Delta(V_i) < \frac{1}{2}\Delta(V_{j_k})$. Let $\gamma(h) = er_{\mathcal{D}}(h) - er_{\mathcal{D}}(h^*)$. By the noise assumption (9) we have that if

$$\gamma(h) \ln \frac{1}{\gamma(h)} \leq c\frac{\Delta(V_{j_k})}{2\theta(\epsilon)}, \tag{10}$$

then $\Delta(V_i) < \frac{1}{2}\Delta(V_{j_k})$. Here and below, $c$ is appropriate constant but may be different from line to line. Note that (10) holds if $\gamma(h) \leq c\frac{\Delta(V_{j_k})}{\theta(\epsilon) \ln \frac{\theta(\epsilon)}{\Delta(V_{j_k})}}$, and in turn if $\gamma(h) \leq c\frac{\Delta(V_{j_k})}{\theta(\epsilon) \ln \frac{1}{\epsilon}}$ since $\Delta(V_{j_k}) \geq \Delta(V_i) > 2\epsilon\theta(\epsilon)$. But to have the last inequality, the algorithm only needs to label $O(\theta^2(\epsilon) \ln^2 \frac{1}{\epsilon} \ln \frac{1}{\delta})$ instances from $DIS(V_i)$. So the total number of labels requested by $A^2$ is $O(\theta^2(\epsilon) \ln \frac{1}{\delta} \ln^3 \frac{1}{\epsilon})$ ∎

Now we give our main label complexity bounds for agnostic active learning.

**Theorem 11** *Let the instance space be $[0, 1]^{d+1}$. Let the Hypothesis space be $\mathcal{H}_C^K$, where $K > d$. Assume that the Bayes classifier $h^*$ of the learning problem is in $\mathcal{H}_C^K$; the noise condition (8) holds; and $\mathcal{D}_{\mathcal{X}}$ has a density bounded by a $K$th order smooth function as in Theorem 6. Then the $A^2$ algorithm outputs $\hat{h}$ with error rate $er_{\mathcal{D}}(\hat{h}) \leq er_{\mathcal{D}}(h^*) + \epsilon$ and the number of labels requested is at most $\tilde{O}\left(\left(\frac{1}{\epsilon}\right)^{\frac{2d}{K+d}} \ln \frac{1}{\delta}\right)$, where in $\tilde{O}$ we hide the polylog $\left(\frac{1}{\epsilon}\right)$ term.*

**Proof** Note that the density $\mathcal{D}_{\mathcal{X}}$ is upper bounded by a smooth function implies that it is also upper bounded by a constant $M$. Combining Theorem 5, 6 and 10 the theorem follows. ∎

Combining Theorem 5, 7 and 10 we can show the following theorem.

**Theorem 12** *Let the instance space be $[0, 1]^{d+1}$. Let the Hypothesis space be $\mathcal{H}_C^\infty$. Assume that the Bayes classifier $h^*$ of the learning problem is in $\mathcal{H}_C^\infty$; the noise condition (8) holds; and $\mathcal{D}_{\mathcal{X}}$ has a density bounded by an infinitely smooth function as in Theorem 7. Then the $A^2$ algorithm outputs $\hat{h}$ with error rate $er_{\mathcal{D}}(\hat{h}) \leq er_{\mathcal{D}}(h^*) + \epsilon$ and the number of labels requested is at most $O\left(polylog\left(\frac{1}{\epsilon}\right) \ln \frac{1}{\delta}\right)$.*

# 4 Conclusion

We show that if the Bayesian classification boundary is smooth and the distribution is bounded by a smooth function, then under some noise condition active learning achieves polynomial or exponential improvement in the label complexity than passive supervised learning according to whether the smoothness is of finite order or infinite.

Although we assume that the classification boundary is the graph of a function, our results can be generalized to the case that the boundaries are a finite number of functions. To be precise, consider $N$ functions $f_1(\mathbf{x}) \leq \cdots \leq f_N(\mathbf{x})$, for all $\mathbf{x} \in [0,1]^d$. Let $f_0(\mathbf{x}) \equiv 0$, $f_{N+1}(\mathbf{x}) \equiv 1$. The positive (or negative) set defined by these functions is $\{(\mathbf{x}, x^{d+1}) : f_{2i}(\mathbf{x}) \leq x \leq f_{2i+1}(\mathbf{x}), \ i = 0, 1, \ldots, \frac{N}{2}\}$. Our theorems still hold in this case. In addition, by techniques in [Dud99] (page 259), our results may generalize to problems which have intrinsic smooth boundaries (not only graphs of functions).

## Appendix

In this appendix we describe very briefly the ideas to prove Lemma 8 and Lemma 9. The formal proofs can be found in the supplementary file.

**Ideas to Prove Lemma 8** First consider the $d = 1$ case. Note that if $f \in F_C^K$, then $|f^{(K-1)}(x) - f^{(K-1)}(x')| \leq C|x - x'|$ for all $x, x' \in [0, 1]$. It is not difficult to see that we only need to show for any $f$ such that $|f^{(K-1)}(x) - f^{(K-1)}(x')| \leq C|x - x'|$, if $\int_0^1 |f(x)|dx = r$, then $\|f\|_\infty = O(r^{\frac{K}{K+1}})$.

To show this, note that in order that $\|f\|_\infty$ achieves the maximum while $\int |f| = r$, the derivatives of $f$ must be as large as possible. Indeed, it can be shown that (one of) the optimal $f$ is of the form

$$f(x) = \begin{cases} \frac{C}{K!}|x - \xi|^K & 0 \leq x \leq \xi, \\ \\ 0 & \xi < x \leq 1. \end{cases} \tag{11}$$

That is, $|f^{(K-1)}(x) - f^{(K-1)}(x')| = C|x - x'|$ (i.e. the $K - 1$ order derivatives reaches the upper bound of the Lipschitz constant.) for all $x, x' \in [0, \xi]$, where $\xi$ is determined by $\int_0^1 f(x)dx = r$. It is then easy to check that $\|f\|_\infty = O(r^{\frac{K}{K+1}})$.

For the general $d > 1$ case, we relax the constraint. Note that all $K - 1$th order partial derivatives are Lipschitz implies that all $K - 1$th order *directional* derivatives are Lipschitz too. Under the latter constraint, (one of) the optimal $f$ has the form

$$f(x) = \begin{cases} \frac{C}{K!}|\|x\| - \xi|^K & 0 \leq \|x\| \leq \xi, \\ \\ 0 & \xi < \|x\|. \end{cases}$$

where $\xi$ is determined by $\int_{[0,1]^d} |f(x)|dx = r$. This implies $\|f\|_\infty = O(r^{\frac{K}{K+d}})$.

**Ideas to Prove Lemma 9** Similar to the proof of Lemma 8, we only need to show that for any $f \in F_C^\infty$, if $\int_{[0,1]^d} |f(x)|dx = r$, then $\|f\|_\infty = O(r \cdot \log^d(\frac{1}{r}))$.

Since $f$ is infinitely smooth, we can choose $K$ large and depending on $r$. For the $d = 1$ case, let $K + 1 = \frac{\log \frac{1}{r}}{\log \log \frac{1}{r}}$. We know that the optimal $f$ is of the form of Eq.(11). (Actually this choice of $K$ is approximately the largest $K$ such that Eq.(11) is still the optimal form. If $K$ is larger than this, $\xi$ will be out of $[0, 1]$.) Since $\int_0^1 |f(x)| = r$, we have $\xi^{K+1} = \frac{(K+1)!}{C}$. Now, $\|f\|_\infty = \frac{C}{K!}\xi^K$. Note that $(\frac{1}{r})^{K+1} = (\frac{1}{r})^{\frac{\log \log \frac{1}{r}}{\log \frac{1}{r}}} = \log \frac{1}{r}$. By Stirling's formula we can show $\|f\|_\infty = O(r \cdot \log \frac{1}{r})$.

For the $d > 1$ case, let $K + d = \frac{\log \frac{1}{r}}{\log \log \frac{1}{r}}$. By similar arguments we can show $\|f\|_\infty = O(r \cdot \log^d \frac{1}{r})$.

## Acknowledgement

This work was supported by NSFC(60775005).

# References

[Ang88]    D. Angluin. Queries and concept learning. *Machine Learning*, 2:319–342, 1988.

[BAL06]    M.-F. Balcan, A.Beygelzimer, and J. Langford. Agnostic active learning. In *23th International Conference on Machine Learning*, 2006.

[CN07]     R. Castro and R. Nowak. Minimax bounds for active learning. In *20th Annual Conference on Learning Theory*, 2007.

[Das05]    S. Dasgupta. Coarse sample complexity bounds for active learning. In *Advances in Neural Information Processing Systems*, 2005.

[DHM07]    S. Dasgupta, D. Hsu, and C. Monteleoni. A general agnostic active learning algorithm. In *Advances in Neural Information Processing Systems*, 2007.

[Dud99]    R.M. Dudley. *Uniform Central Limit Theorems*. Cambridge University Press, 1999.

[Fri09]    E. Friedman. Active learning for smooth problems. In *22th Annual Conference on Learning Theory*, 2009.

[GKW03]    V.E. Gine, V.I. Koltchinskii, and J. Wellner. Ratio limit theorems for empirical processes. *Stochastic Inequalities and Applications*, 56:249–278, 2003.

[Han07]    S. Hanneke. A bound on the label complexity of agnostic active learning. In *24th International Conference on Machine Learning*, 2007.

[Han09]    S. Hanneke. Adaptive rates of convergence in active learning. In *22th Annual Conference on Learning Theory*, 2009.

[Kaa06]    M. Kaariainen. Active learning in the non-realizable case. In *17th International Conference on Algorithmic Learning Theory*, 2006.

[Tsy04]    A. Tsybakov. Optimal aggregation of classifiers in statistical learning. *The Annals of Statistics*, 32:135–166, 2004.

[vdG00]    S. van de Geer. *Applications of Empirical Process Theory*. Cambridge University Press, 2000.

[vdVW96]   A. van der Vaart and J. Wellner. *Weak Convergence and Empirical Processes with Application to Statistics*. Springer Verlag, 1996.

